# Regularized Learning with Networks of Features

**Ted Sandler, Partha Pratim Talukdar, and Lyle H. Ungar**
Department of Computer & Information Science, University of Pennsylvania
{tsandler,partha,ungar}@cis.upenn.edu

**John Blitzer**
Department of Computer Science, U.C. Berkeley
blitzer@cs.berkeley.edu

## Abstract

For many supervised learning problems, we possess prior knowledge about which features yield similar information about the target variable. In predicting the topic of a document, we might know that two words are synonyms, and when performing image recognition, we know which pixels are adjacent. Such synonymous or neighboring features are near-duplicates and should be expected to have similar weights in an accurate model. Here we present a framework for regularized learning when one has prior knowledge about which features are expected to have similar and dissimilar weights. The prior knowledge is encoded as a network whose vertices are features and whose edges represent similarities and dissimilarities between them. During learning, each feature's weight is penalized by the amount it differs from the average weight of its neighbors. For text classification, regularization using networks of word co-occurrences outperforms manifold learning and compares favorably to other recently proposed semi-supervised learning methods. For sentiment analysis, feature networks constructed from declarative human knowledge significantly improve prediction accuracy.

## 1 Introduction

For many important problems in machine learning, we have a limited amount of labeled training data and a very high-dimensional feature space. A common approach to alleviating the difficulty of learning in these settings is to regularize a model by penalizing a norm of its parameter vector. The most commonly used norms in classification, $L_1$ and $L_2$, assume independence among model parameters [1]. However, we often have access to information about dependencies between parameters. For example, with spatio-temporal data, we usually know which measurements were taken at points nearby in space and time. And in natural language processing, digital lexicons such as Word-Net can indicate which words are synonyms or antonyms [2]. For the biomedical domain, databases such as KEGG and DIP list putative protein interactions [3, 4]. And in the case of semi-supervised learning, dependencies can be inferred from unlabeled data [5, 6]. Consequently, we should be able to learn models more effectively if we can incorporate dependency structure directly into the norm used for regularization.

Here we introduce regularized learning with networks of features, a framework for constructing customized norms on the parameters of a model when we have prior knowledge about which parameters are likely to have similar values. Since our focus is on classification, the parameters we consider are feature weights in a linear classifier. The prior knowledge is encoded as a network or graph whose nodes represent features and whose edges represent similarities between the features in terms of how likely they are to have similar weights. During learning, each feature's weight is penalized by the amount it differs from the average weight of its neighbors. This regularization objective is closely

connected to the unsupervised dimensionality reduction method, locally linear embedding (LLE), proposed by Roweis and Saul [7]. In LLE, each data instance is assumed to be a linear combination of its nearest neighbors on a low dimensional manifold. In this work, each feature's weight is preferred (though not required) to be a linear combination of the weights of its neighbors.

Similar to other recent methods for incorporating prior knowledge in learning, our framework can be viewed as constructing a Gaussian prior with non-diagonal covariance matrix on the model parameters [6, 8]. However, instead of constructing the covariance matrix directly, it is induced from a network. The network is typically sparse in that each feature has only a small number of neighbors. However, the induced covariance matrix is generally dense. Consequently, we can implicitly construct rich and dense covariance matrices over large feature spaces without incurring the space and computational blow-ups that would be incurred if we attempted to construct these matrices explicitly.

Regularization using networks of features is especially appropriate for high-dimensional feature spaces such as are encountered in text processing where the local distances required by traditional manifold classification methods [9, 10] may be difficult to estimate accurately, even with large amounts of unlabeled data. We show that regularization with feature-networks derived from word co-occurrence statistics outperforms manifold regularization and another, more recent, semi-supervised learning approach [5] on the task of text classification. Feature network based regularization also supports extensions which provide flexibility in modeling parameter dependencies, allowing for feature dissimilarities and the introduction of feature classes whose weights have common but unknown means. We demonstrate that these extensions improve classification accuracy on the task of classifying product reviews in terms of how favorable they are to the products in question [11]. Finally, we contrast our approach with related regularization methods.

## 2   Regularized Learning with Networks of Features

We assume a standard supervised learning framework in which we are given a training set of instances $T = \{(\mathbf{x}_i, y_i)\}_{i=1}^n$ with $\mathbf{x}_i \in \mathbb{R}^d$ and associated labels $y_i \in \mathcal{Y}$. We wish to learn a linear classifier parameterized by weight vector $\mathbf{w} \in \mathbb{R}^d$ by minimizing a convex loss function $l(\mathbf{x}, y \,; \mathbf{w})$ over the training instances, $(\mathbf{x}_i, y_i)$. For many problems, the dimension, $d$, is much larger than the number of labeled instances, $n$. Therefore, it is important to impose some constraints on $\mathbf{w}$. Here we do this using a directed network or graph, $G$, whose vertices, $V = \{1, ..., d\}$, correspond to the features of our model and whose edges link features whose weights are believed to be similar. The edges of $G$ are non-negative with larger weights indicating greater similarity. Conversely, a weight of zero means that two features are not believed *a priori* to be similar. As has been shown elsewhere [5, 6, 8], such similarities can be inferred from prior domain knowledge, auxiliary task learning, and statistics computed on unlabeled data. For the time being we assume that $G$ is given and defer its construction until section 4, experimental work.

The weights of $G$ are encoded by a matrix, $P$, where $P_{ij} \geq 0$ gives the weight of the directed edge from vertex $i$ to vertex $j$. We constrain the out-degree of each vertex to sum to one, $\sum_j P_{ij} = 1$, so that no feature "dominates" the graph. Because the semantics of the graph are that linked features should have similar weights, we penalize each feature's weight by the squared amount it differs from the weighted average of its neighbors. This gives us the following criterion to optimize in learning:

$$\text{loss}(\mathbf{w}) = \sum_{i=1}^n l(\mathbf{x}_i, y_i \,; \mathbf{w}) + \alpha \sum_{j=1}^d \big(\mathbf{w}_j - \sum_k P_{jk}\, \mathbf{w}_k\big)^2 + \beta \,\|\mathbf{w}\|_2^2, \tag{1}$$

where we have added a ridge term to make the loss strictly convex. The hyperparameters $\alpha$ and $\beta$ specify the amount of network and ridge regularization respectively. The regularization penalty can be rewritten as $\mathbf{w}^\top M \mathbf{w}$ where $M = \alpha\,(I-P)^\top (I-P) + \beta\, I$. The matrix $M$ is symmetric positive definite, and therefore our criterion possesses a Bayesian interpretation in which the weight vector, $\mathbf{w}$, is *a priori* normally distributed with mean zero and covariance matrix $2M^{-1}$.

Minimizing equation (1) is equivalent to finding the MAP estimate for $\mathbf{w}$. The gradient of (1) with respect to $\mathbf{w}$ is $\nabla_{\mathbf{w}} \text{loss} = \sum_{i=1}^n \nabla_{\mathbf{w}} l(\mathbf{x}_i, y_i \,; \mathbf{w}) + 2M\mathbf{w}$ and therefore requires only an additional matrix multiply on top of computing the loss over the training data. If $P$ is sparse, as it is in our experiments—i.e., it has only $kd$ entries for $k \ll d$—then the matrix multiply is $O(d)$. Thus

equation (1) can be minimized very quickly. Additionally, the induced covariance matrix $M^{-1}$ will typically be dense even though $P$ is sparse, showing that we can construct dense covariance structures over $\mathbf{w}$ without incurring storage and computation costs.

## 2.1 Relationship to Locally Linear Embedding

Locally linear embedding (LLE) is an unsupervised learning method for embedding high dimensional data in a low dimensional vector space. The data $\{\vec{X}_i\}_{i=1}^n$ is assumed to lie on a low dimensional manifold of dimension $c$ within a high dimensional vector space of dimension $d$ with $c \ll d$. Since the data lies on a manifold, each point is approximately a convex combination of its nearest neighbors on the manifold. That is, $\vec{X}_i \approx \sum_{j \sim i} P_{ij} \vec{X}_j$, where $j \sim i$ denotes the samples, $j$, which lie close to $i$ on the manifold. As above, the matrix $P$ has non-negative entries and its rows sum to one. The set of low dimensional coordinates, $\{\vec{Y}_i\}_{i=1}^n$, $\vec{Y}_i \in \mathbb{R}^c$, are found by minimizing the sum of squares cost:

$$\text{cost}(\{\vec{Y}_i\}) = \sum_i \|\vec{Y}_i - \sum_j P_{ij}\vec{Y}_j\|_2^2, \tag{2}$$

subject to the constraint that the $\{\vec{Y}_i\}$ have unit variance in each of the $c$ dimensions. The solution to equation (2) is found by performing eigen-decomposition on the matrix $(I - P)^\top (I - P) = U\Lambda U^\top$ where $U$ is the matrix of eigenvectors and $\Lambda$ is the diagonal matrix of eigenvalues. The LLE coordinates are obtained from the eigenvectors, $u_1, ..., u_c$ whose eigenvalues, $\lambda_1, ..., \lambda_c$, are smallest[1] by setting $\vec{Y}_i = (u_{1i}, ..., u_{ci})^\top$. Looking at equation (1) and ignoring the ridge term, it is clear that our feature network regularization penalty is identical to LLE except that the embedding is found for the feature weights rather than data instances. However, there is a deeper connection.

If we let $L(Y, X\mathbf{w})$ denote the unregularized loss over the training set where $X$ is the $n \times d$ matrix of instances and $Y$ is the $n$-vector of class labels, we can express equation (1) in matrix form as

$$\mathbf{w}^* = \underset{\mathbf{w}}{\text{argmin}}\, L(Y, X\mathbf{w}) + \mathbf{w}^\top \big( \alpha\, (I - P)^\top (I - P) + \beta\, I \big)\, \mathbf{w}. \tag{3}$$

Defining $\tilde{X}$ to be $XU(\alpha\Lambda + \beta I)^{-1/2}$ where $U$ and $\Lambda$ are from the eigen-decomposition above, it is not hard to show that equation (3) is equivalent to the alternative ridge regularized learning problem

$$\tilde{\mathbf{w}}^* = \underset{\tilde{\mathbf{w}}}{\text{argmin}}\, L(Y, \tilde{X}\tilde{\mathbf{w}}) + \tilde{\mathbf{w}}^\top \tilde{\mathbf{w}}. \tag{4}$$

That is, the two minimizers, $\mathbf{w}$ and $\tilde{\mathbf{w}}$, yield the same predictions: $\hat{Y} = X\mathbf{w} = \tilde{X}\tilde{\mathbf{w}}$. Consequently, we can view feature network regularization as: 1) finding an embedding for the features using LLE in which all of the eigenvectors are used and scaled by the inverse square-roots of their eigenvalues (plus a smoothing term, $\beta I$, that makes the inverse well-defined); 2) projecting the data instances onto these coordinates; and 3) learning a ridge-penalized model for the new representation. In using all of the eigenvectors, the dimensionality of the feature embedding is not reduced. However, in scaling the eigenvectors by the inverse square-roots of their eigenvalues, the directions of least cost in the network regularized problem become the directions of maximum variance in the associated ridge regularized problem, and hence are the directions of least cost in the ridge problem. As a result, the effective dimensionality of the learning problem is reduced to the extent that the distribution of inverted eigenvalues is sharply peaked. When the best representation for classification has high dimension, it is faster to solve (3) than to compute a large eigenvector basis and solve (4). In the high dimensional problems of section 4, we find that regularization with feature networks outperforms LLE-based regression.

## 3 Extensions to Feature Network Regularization

In this section, we pose a number of extensions and alternatives to feature network regularization as formulated in section 2, including the modeling of classes of features whose weights are believed to share the same unknown means, the incorporation of feature dissimilarities, and two alternative regularization criteria based on the graph Laplacian.

### 3.1 Regularizing with Classes of Features

In machine learning, features can often be grouped into classes, such that all the weights of the features in a given class are drawn from the same underlying distribution. For example, words can be grouped by part of speech, by meaning (as in WordNet's synsets), or by clustering based on the words they co-occur with or the documents they occur in. Using an appropriately constructed feature graph, we can model the case in which the underlying distributions are believed to be Gaussians with known, identical variances but with unknown means. That is, the case in which there are $k$ disjoint classes of features $\{C_i\}_{i=1}^k$ whose weights are drawn i.i.d. $N(\mu_i, \sigma^2)$ with $\mu_i$ unknown but $\sigma^2$ known and shared across all classes.

The straight-forward approach to modeling this scenario might seem to be to link all the features within a class to each other, forming a clique, but this does not lead to the desired interpretation. Additionally, the number of edges in this construction scales quadratically in the clique sizes, resulting in feature graphs that are not sparse. Our approach is therefore to create $k$ additional "virtual" features, $f_1, ..., f_k$, that do not appear in any of the data instances but whose weights $\hat{\mu}_1, ..., \hat{\mu}_k$ serve as the estimates for the true but unknown means, $\mu_1, ..., \mu_k$. In creating the feature graph, we link each feature to the virtual feature for its class with an edge of weight one. The virtual features, themselves, do not possess any out-going links.

Denoting the class of feature $i$ as $c(i)$, and setting the hyperparameters $\alpha$ and $\beta$ in equation (1) to $1/(2\sigma^2)$ and 0, respectively, yields a network regularization cost of $\frac{1}{2}\sigma^{-2}\sum_{i=1}^d (\mathbf{w}_i - \hat{\mu}_{c(i)})^2$. Since the virtual features do not appear in any instances, i.e. their values are zero in every data instance, their weights are free to take on whatever values minimize the network regularization cost in (1), in particular the estimates of the class means, $\mu_1, ..., \mu_k$. Consequently, minimizing the network regularization penalty maximizes the log-likelihood for the intended scenario. We can extend this construction to model the case in which the feature weights are drawn from a mixture of Gaussians by connecting each feature to a number of virtual features with edge weights that sum to one.

### 3.2 Incorporating Feature Dissimilarities

Feature network regularization can also be extended to induce features to have opposing weights. Such feature "dissimilarities" can be useful in tasks such as sentiment prediction where we would like weights for words such as "great" or "fantastic" to have opposite signs from their negated bigram counterparts "not great" and "not fantastic," and from their antonyms. To model dissimilarities, we construct a separate graph whose edges represent anti-correlations between features. Regularizing over this graph enforces each feature's weight to be equal to the negative of the average of the neighboring weights. To do this, we encode the dissimilarity graph using a matrix $Q$, defined analogously to the matrix $P$, and add the term $\sum_i \left(\mathbf{w}_i + \sum_j Q_{ij}\,\mathbf{w}_j\right)^2$ to the network regularization criterion, which can be written as $\mathbf{w}^\top (I+Q)^\top (I+Q)\mathbf{w}$. The matrix $(I+Q)^\top (I+Q)$ is positive semidefinite like its similarity graph counterpart. Goldberg et al. [12] use a similar construction with the graph Laplacian in order to incorporate dissimilarities between instances in manifold learning.

### 3.3 Regularizing Features with the Graph Laplacian

A natural alternative to the network regularization criterion given in section (2) is to regularize the feature weights using a penalty derived from the graph Laplacian [13]. Here, the feature graph's edge weights are given by a symmetric matrix, $W$, whose entries, $W_{ij} \geq 0$, give the weight of the edge between features $i$ and $j$. The Laplacian penalty is $\frac{1}{2}\sum_{i,j} W_{ij}(\mathbf{w}_i - \mathbf{w}_j)^2$ which can be written as $\mathbf{w}^\top (D-W)\,\mathbf{w}$, where $D = \mathrm{diag}(W\mathbf{1})$ is the vertex degree matrix. The main difference between the Laplacian penalty and the network penalty in equation (1) is that the Laplacian penalizes each edge equally (modulo the edge weights) whereas the network penalty penalizes each feature equally. In graphs where there are large differences in vertex degree, the Laplacian penalty will therefore focus most of the regularization cost on features with many neighbors. Experiments in section 4 show that the criterion in (1) outperforms the Laplacian penalty as well as a related penalty derived from the normalized graph Laplacian, $\frac{1}{2}\sum_{i,j} W_{ij}(\mathbf{w}_i/\sqrt{D_{ii}} - \mathbf{w}_j/\sqrt{D_{jj}})^2$. The normalized Laplacian penalty assumes that $\sqrt{D_{jj}}\mathbf{w}_i \approx \sqrt{D_{ii}}\mathbf{w}_j$, which is different from assuming that linked features should have similar weights.

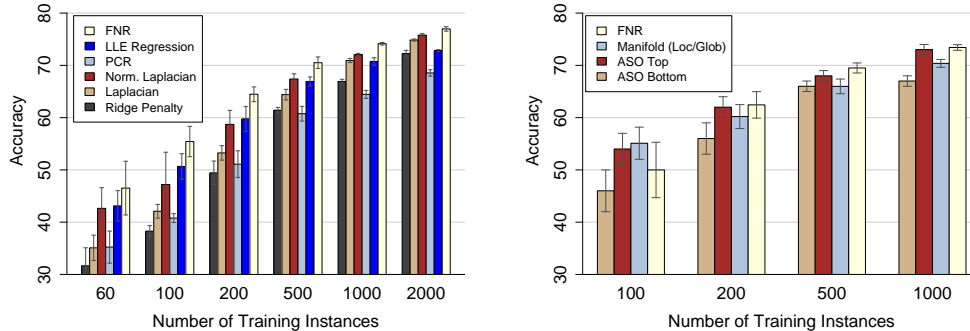

Figure 1: **Left:** Accuracy of feature network regularization (FNR) and five baselines on "20 newsgroups" data. **Right:** Accuracy of FNR compared to reported accuracies of three other semi-supervised learning methods.

## 4  Experiments

We evaluated logistic regression augmented with feature network regularization on two natural language processing tasks. The first was document classification on the 20 Newsgroups dataset, a well-known document classification benchmark. The second was sentiment classification of product reviews, the task of classifying user-written reviews according to whether they are favorable or unfavorable to the product under review based on the review text [11]. Feature graphs for the two tasks were constructed using different information. For document classification, the feature graph was constructed using feature co-occurrence statistics gleaned from unlabeled data. In sentiment prediction, both co-occurrence statistics and prior domain knowledge were used.

### 4.1  Experiments on 20 Newsgroups

We evaluated feature network based regularization on the 20 newsgroups classification task using all twenty classes. The feature set was restricted to the 11,376 words which occurred in at least 20 documents, not counting stop-words. Word counts were transformed by adding one and taking logs. To construct the feature graph, each feature (word) was represented by a binary vector denoting its presence/absence in each of the 20,000 documents of the dataset. To measure similarity between features, we computed cosines between these binary vectors. Each feature was linked to the 25 other features with highest cosine scores, provided that the scores were above a minimum threshold of 0.10. The edge weights of the graph were set to these cosine scores and the matrix $P$ was constructed by normalizing each vertex's out-degree to sum to one.

Figure 1 (left) shows feature network regularization compared against five other baselines: logistic regression with an $L_2$ (ridge) penalty; principal components logistic regression (PCR) in which each instance was projected onto the largest 200 right singular vectors of the $n \times d$ matrix, $X$; LLE-logistic regression in which each instance was projected onto the smallest 200 eigenvectors of the matrix $(I-P)^\top(I-P)$ described in section 2; and logistic regression regularized by the normalized and unnormalized graph Laplacians described in section 3.3. Results at each training set size are averages of five trials with training sets sampled to contain an equal number of documents per class. For ridge, the amount of $L_2$ regularization was chosen using cross validation on the training set. Similarly, for feature network regularization and the Laplacian regularizers, the hyperparameters $\alpha$ and $\beta$ were chosen through cross validation on the training set using a simple grid search. The ratio of $\alpha$ to $\beta$ tended to be around 100:1. For PCR and LLE-logistic regression, the number of eigenvectors used was chosen to give good performance on the test set at both large and small training set sizes. All models were trained using L-BFGS with a maximum of 200 iterations. Learning a single model took between between 30 seconds and two minutes, with convergence typically achieved before the full 200 iterations.

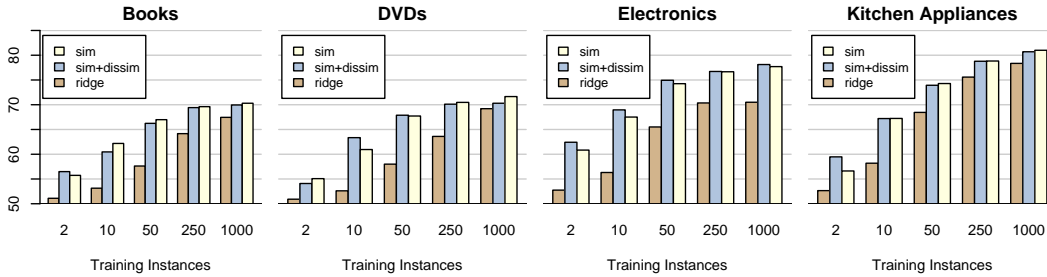

Figure 2: Accuracy of feature network regularization on the sentiment datasets using feature classes and dissimilarity edges to regularize the small sent of SentiWordNet features.

The results in figure 1 show that feature network regularization with a graph constructed from unlabeled data outperforms all baselines and increases accuracy by 4%-17% over the plain ridge penalty, an error reduction of 17%-30%. Additionally, it outperforms the related LLE regression. We conjecture this is because in tuning the hyperparameters, we can adaptively tune the dimensionality of the underlying data representation. Additionally, by scaling the eigenvectors by their eigenvalues, feature network regularization keeps more information about the directions of least cost in weight space than does LLE regression, which does not rescale the eigenvectors but simply keeps or discards them (i.e. scales them by 1 or 0).

Figure 1 (right) compares feature network regularization against two external approaches that leverage unlabeled data: a multi-task learning approach called alternating structure optimization (ASO), and our reimplementation of a manifold learning method which we refer to as "local/global consistency" [5, 10]. To make a fair comparison against the reported results for ASO, training sets were sampled so as not to necessarily contain an equal number of documents per class. Accuracies are given for the highest and lowest performing variants of ASO reported in [5]. Our reimplementation of local/global consistency used the same document preprocessing described in [10]. However, the graph was constructed so that each document had only $K = 10$ neighbors (the authors in [10] use a fully connected graph which does not fit in memory for the entire 20 newsgroups dataset). Classification accuracy of local/global consistency did not vary much with $K$ and up to 500 neighbors were tried for each document. Here we see that feature network regularization is competitive with the other semi-supervised methods and performs best at all but the smallest training set size.

## 4.2  Sentiment Classification

For sentiment prediction, we obtained the product review datasets used in [11]. Each dataset consists of reviews downloaded from Amazon.com for one of four different product domains: books, DVDs, electronics, and kitchen appliances. The reviews have an associated number of "stars," ranging from 0 to 5, rating the quality of a product. The goal of the task is to predict whether a review has more than (positive) or less than (negative) 3 stars associated with it based only on the text in the review. We performed two sets of experiments in which prior domain knowledge was incorporated using feature networks. In both, we used a list of sentimentally-charged words obtained from the SentiWordNet database [14], a database which associates positive and negative sentiment scores to each word in WordNet. In the first experiment, we constructed a set of feature classes in the manner described in section 3.1 to see if such classes could be used to boot-strap weight polarities for groups of features. In the second, we computed similarities between words in terms of the similarity of their co-occurrence's with the sentimentally charged words.

From SentiWordNet we extracted a list of roughly 200 words with high positive and negative sentiment scores that also occurred in the product reviews at least 100 times. Words to which SentiWordNet gave a high 'positive' score were placed in a "positive words" cluster and words given a high 'negative' score were placed in a "negative words" cluster. As described in section 3.1, all words in the positive cluster were attached to a virtual feature representing the mean feature weight of the positive cluster words, and all words in the negative cluster were attached to a virtual weight representing the mean weight of the negative cluster words. We also added a dissimilarity edge (described in section 3.2) between the positive and negative clusters' virtual features to induce the two

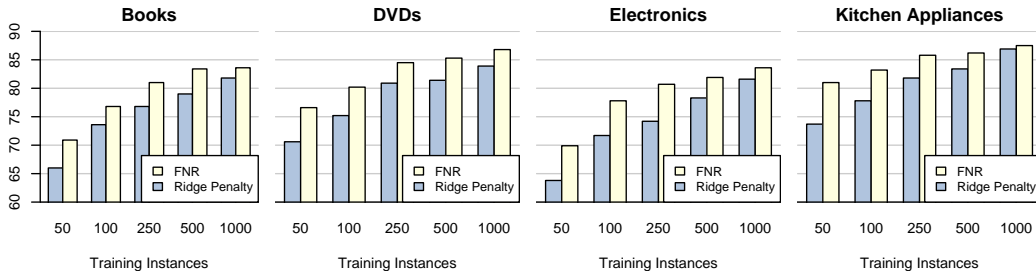

Figure 3: Accuracy of feature network and ridge regularization on four sentiment classification datasets.

classes of features to have opposite means. As shown in figure 2, imposing feature clusters on the two classes of words improves performance noticeably while the addition of the feature dissimilarity edge does not yield much benefit. When it helps, it is only for the smallest training set sizes.

This simple set of experiments demonstrated the applicability of feature classes for inducing groups of features to have similar means, and that the words extracted from SentiWordNet were relatively helpful in determining the sentiment of a review. However, the number of features used in these experiments was too small to yield reasonable performance in an applied setting. Thus we extended the feature sets to include all unigram and bigram word-features which occurred in ten or more reviews. The total number of reviews and size of the feature sets is given in table 1.

The method used to construct the feature graph in the 20 newsgroups experiments was not well suited for sentiment prediction since plain feature co-occurrence statistics tended to find groups of words that showed up in reviews for products of the same type, e.g., digital cameras or laptops. While such similarities are useful in predicting what type of product is being reviewed, they are of little help in determining whether a review is favorable or unfavorable. Thus, to align features along dimensions

| Dataset | Instances | Features | Edges |
|---|---|---|---|
| books | 13,161 | 29,404 | 470,034 |
| DVDs | 13,005 | 31,475 | 419,178 |
| electronics | 8,922 | 15,104 | 343,890 |
| kitchen | 7,760 | 11,658 | 305,926 |

Table 1: Sentiment Data Statistics

of 'sentiment,' we computed the correlations of all features with the SentiWordNet features so that each word was represented as a 200 dimensional vector of correlations with these highly charged sentiment words. Distances between these correlation vectors were computed in order to determine which features should be linked. We next computed each feature's 100 nearest neighbors. Two features were linked if both were in the other's set of nearest 100 neighbors. For simplicity, the edge weights were set to one and the graph weight matrix was then row-normalized in order to construct the matrix $P$. The number of edges in each feature graph is given in table 1.

The 'kitchen' dataset was used as a development dataset in order to arrive at the method for constructing the feature graph and for choosing the hyperparameter values: $\alpha = 9.9$ and $\beta = 0.1$. Figure 3 gives accuracy results for all four sentiment datasets at training sets of 50 to 1000 instances. The results show that linking features which are similarly correlated with sentiment-loaded words yields improvements on every dataset and at every training set size.

## 5  Related Work

Most similar to the work presented here is that of the fused lasso (Tibshirani et al. [15]) which can be interpreted as using the graph Laplacian regularizer but with an $L_1$ norm instead of $L_2$ on the residuals of weight differences: $\sum_i \sum_{j \sim i} |\mathbf{w}_i - \mathbf{w}_j|$ and all edge weights set to one. As the authors discuss, an $L_1$ penalty prefers that weights of linked features be exactly equal so that the residual vector of weight differences is sparse. $L_1$ is appropriate if the true weights are believed to be exactly equal, but in many settings, features are *near* copies of one another whose weights should be similar rather than identical. Thus in these settings, penalizing squared differences rather than absolute ones is more appropriate. Optimizing $L_1$ feature weight differences also leads to a much harder optimization problem, making it less applicable in large scale learning. Li and Li [13] regularize

feature weights using the normalized graph Laplacian in their work on biomedical prediction tasks. As shown, this criterion does not work as well on the text prediction problems considered here.

Krupka and Tishby [8] proposed a method for inducing feature-weight covariance matrices using distances in a "meta-feature" space. Under their framework, two features positively covary if they are close in this space and approach independence as they grow distant. The authors represent each feature $i$ as a vector of meta-features, $\mathbf{u}_i$, and compute the entries of the feature weight covariance matrix, $C_{ij} = \exp(-\frac{1}{2\sigma^2}\|\mathbf{u}_i - \mathbf{u}_j\|^2)$. Obviously, the choice of which is more appropriate, a feature graph or metric space, is application dependent. However, it is less obvious how to incorporate feature dissimilarities in a metric space. A second difference is that our work defines the regularizer in terms of $C^{-1} \approx (I-P)^\top (I-P)$ rather than $C$ itself. While $C^{-1}$ is constructed to be sparse with a nearest neighbors graph, the induced covariance matrix, $C$, need not be sparse. Thus, working with $C^{-1}$ allows for construct dense covariance matrices without having to explicitly store them. Finally, Raina et al. [6] learn a feature-weight covariance matrix via auxiliary task learning. Interestingly, the entries of this covariance matrix are learned jointly with a regression model for predicting feature weight covariances as a function of meta-features. However, since their approach explicitly predicts each entry of the covariance matrix, they are restricted to learning smaller models, consisting of hundreds rather than tens of thousands of features.

# 6    Conclusion

We have presented regularized learning with networks of features, a simple and flexible framework for incorporating expectations about feature weight similarities in learning. Feature similarities are modeled using a feature graph and the weight of each feature is preferred to be close to the average of its neighbors. On the task of document classification, feature network regularization is superior to several related criteria, as well as to a manifold learning approach where the graph models similarities between instances rather than between features. Extensions for modeling feature classes, as well as feature dissimilarities, yielded benefits on the problem of sentiment prediction.

## Footnotes

[1]More precisely, eigenvectors $u_2, ..., u_{c+1}$ are used so that the $\{\vec{Y}_i\}$ are centered.

## References

[1]  T. Hastie, R. Tibshirani, and J. Friedman. *The Elements of Statistical Learning*. Springer New York, 2001.

[2]  C. Fellbaum. *WordNet: an electronic lexical database*. MIT Press, 1998.

[3]  H. Ogata, S. Goto, K. Sato, W. Fujibuchi, H. Bono, and M. Kanehisa. KEGG: Kyoto Encyclopedia of Genes and Genomes. *Nucleic Acids Research*, 27(1):29–34, 1999.

[4]  I. Xenarios, D.W. Rice, L. Salwinski, M.K. Baron, E.M. Marcotte, and D. Eisenberg. DIP: The Database of Interacting Proteins. *Nucleic Acids Research*, 28(1):289–291, 2000.

[5]  R.K. Ando and T. Zhang. A Framework for Learning Predictive Structures from Multiple Tasks and Unlabeled Data. *JMLR*, 6:1817–1853, 2005.

[6]  R. Raina, A.Y. Ng, and D. Koller. Constructing informative priors using transfer learning. In *ICML*, 2006.

[7]  S.T. Roweis and L.K. Saul. Nonlinear Dimensionality Reduction by Locally Linear Embedding. *Science*, 290(5500):2323–2326, 2000.

[8]  E. Krupka and N. Tishby. Incorporating Prior Knowledge on Features into Learning. In *AISTATS*, 2007.

[9]  M. Belkin, P. Niyogi, and V. Sindhwani. Manifold regularization: a geometric framework for lerning from lableed and unlabeled examples. *JMLR*, 7:2399–2434, 2006.

[10]  D. Zhou, O. Bousquet, T.N. Lal, J. Weston, and B. Schölkopf. Learning with local and global consistency. In *NIPS*, 2004.

[11]  J. Blitzer, M. Dredze, and F. Pereira. Biographies, Bollywood, Boom-boxes and Blenders: Domain Adaptation for Sentiment Classification. In *ACL*, 2007.

[12]  A.B. Goldberg, X. Zhu, and S. Wright. Dissimilarity in Graph-Based Semi-Supervised Classification. In *AISTATS*, 2007.

[13]  C. Li and H. Li. Network-constrained regularization and variable selection for analysis of genomic data. *Bioinformatics*, 24(9):1175–1182, 2008.

[14]  A. Esuli and F. Sebastiani. SentiWordNet: A Publicly Available Lexical Resource For Opinion Mining. In *LREC*, 2006.

[15]  R. Tibshirani, M. Saunders, S. Rosset, J. Zhu, and K. Knight. Sparsity and Smoothness via the Fused Lasso. *Journal of the Royal Statistical Society Series B*, 67(1):91–108, 2005.
